# Structured Ranking Learning using Cumulative Distribution Networks

**Jim C. Huang**
Probabilistic and Statistical Inference Group
University of Toronto
Toronto, ON, Canada M5S 3G4
`jim@psi.toronto.edu`

**Brendan J. Frey**
Probabilistic and Statistical Inference Group
University of Toronto
Toronto, ON, Canada M5S 3G4
`frey@psi.toronto.edu`

## Abstract

Ranking is at the heart of many information retrieval applications. Unlike standard regression or classification in which we predict outputs independently, in ranking we are interested in predicting structured outputs so that misranking one object can significantly affect whether we correctly rank the other objects. In practice, the problem of ranking involves a large number of objects to be ranked and either approximate structured prediction methods are required, or assumptions of independence between object scores must be made in order to make the problem tractable. We present a probabilistic method for learning to rank using the graphical modelling framework of cumulative distribution networks (CDNs), where we can take into account the structure inherent to the problem of ranking by modelling the joint cumulative distribution functions (CDFs) over multiple pairwise preferences. We apply our framework to the problem of document retrieval in the case of the OHSUMED benchmark dataset. We will show that the RankNet, ListNet and ListMLE probabilistic models can be viewed as particular instances of CDNs and that our proposed framework allows for the exploration of a broad class of flexible structured loss functionals for learning to rank.

## 1   Introduction

Ranking is the central problem for many information retrieval applications such as web search, collaborative filtering and document retrieval [8]. In these problems, we are given a set of objects to be ranked and a series of observations where each observation consists of some subset of the objects, a feature vector and some ordering of the objects with highly ranked objects corresponding to a higher relevance or degree of importance. The goal is to then learn a model which allows us to assign a score to new test objects: this often takes the form of a ranking function [2, 4] which assigns a higher score to objects with higher rankings. Unlike the canonical problems of regression or classification in which we predict outputs independently of one another, in ranking we are interested in predicting structured outputs, as the rank of one item can only be determined given the scores of all other items, and so complex inter-dependencies exist between outputs. This requires measures of loss which are multivariate and structured. However, such ranking measures are typically difficult to optimize directly [3], making the problem of learning difficult. A previous approach has been to treat the problem as one of structured prediction [7], where the aim is to directly optimize ranking measures. Another approach has been to approximate these ranking measures with smooth differentiable loss functionals by formulating probabilistic models on pairwise preferences between objects (RankNet; [2]), or on ordered lists of objects (ListNet and ListMLE; [4, 13]). In practice, these methods either require approximating a learning problem with an intractable number of constraints, or they require observations containing complete orderings over the objects to be ranked or one must make independence assumptions on pairwise preferences.

In practice however, we can take advantage of the fact that each observation in the training set only provides preference information about a small subset of the objects to be ranked, so that a sensible probabilistic representation would be the probability of observing a partial ordering over

nodes for a given observation. We will show that 1) a probability over orderings is equivalent to a probability over pairwise inequalities between objects to be ranked and 2) this amounts to specifying a joint cumulative distribution function (CDF) over pairwise object preferences. We will present a framework for ranking using the recently-developed probabilistic graphical modelling framework of CDNs which compactly represents this joint CDF as a product of local functions [5]. While the problem of inference in CDNs was addressed in [5], here we address the problem of learning in CDNs in the context of ranking learning where we estimate model parameters under a structured loss functional that accounts for dependencies between pairwise object preferences. We will then test the proposed framework on the OHSUMED dataset [8], a benchmark dataset used in information retrieval research. Finally we will show that the frameworks proposed by [2, 4, 13] can be viewed as particular types of CDNs so that novel classes of flexible structured loss functionals for ranking learning can be specified under our framework.

## 2  Cumulative distribution networks

The CDN [5] is an undirected graphical model in which the joint CDF $F(\mathbf{z})$ over a set of random variables is represented as a product over functions defined over subsets of these variables. More formally,

$$F(\mathbf{z}) = \prod_{c \in \mathcal{C}} \phi_c(\mathbf{z}_c), \tag{1}$$

where $\phi_c(\mathbf{z}_c)$ is a function defined over some subset of variables. An example of a CDN is shown in Figure 1(a), along with an example bivariate density which can be obtained by differentiating a product of 2 Gaussian CDF functions (Figure 1(b)).

In contrast to undirected models for probability density functions, the global normalization constraint on the CDF does not require computing a partition function and can be enforced locally for each $\phi_c(\mathbf{z}_c)$. Thus, in order for the CDN to represent a valid CDF, it is sufficient that each of the local functions $\phi_c$ satisfy all of the properties of a multivariate CDF. These properties include the requirements that each CDN function $\phi_c$ be bounded between 0 and 1, and that each $\phi_c$ is monotonically non-decreasing with respect to all of its argument variables $\mathbf{z}_c$, so that the joint CDF $F(\mathbf{z})$ is also bounded between 0 and 1 and is monotonically non-decreasing with respect to any and all subsets of variables. In a CDN, disjoint sets of variables $A, B$ are marginally independent if they share no functions in common, and disjoint sets of variables $A, B$ are conditionally independent given variable set $C$ if no path linking any variable in $A$ to any variable in $B$ passes through $C$. In addition, marginalization of variables in a CDN can be done in constant-time via a trivial maximization of the joint CDF with respect to the variables being marginalized. The problem of inference in a CDN can be solved efficiently using a message-passing algorithm called derivative-sum-product. For detailed derivations of the properties of CDNs, including marginal and conditional independence properties, we refer the reader to [5]. The CDN framework provides us with a means to compactly represent multivariate joint CDFs over many variables: in the next section we will formulate a loss functional for learning to rank which takes on such a form.

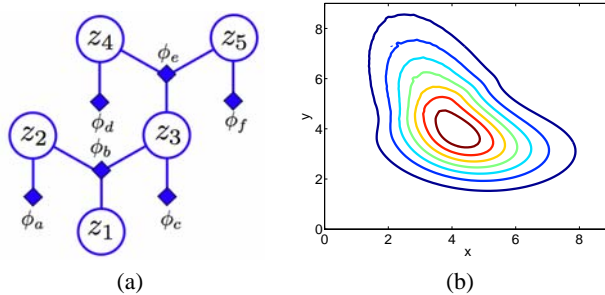

(a)                                                (b)

Figure 1:  a) Cumulative distribution network representing the joint CDF $F(z_1, z_2, z_3, z_4, z_5) = \phi_a(z_2)\phi_b(z_1, z_2, z_3)\phi_c(z_3)\phi_d(z_4)\phi_e(z_3, z_4, z_5)\phi_f(z_5)$; b) Example of a bivariate density $P(x, y)$ corresponding to differentiating a CDF $F(x, y)$ obtained from taking the product of 2 Gaussian bivariate CDFs.

# 3 Structured loss functionals for ranking learning

We now proceed to formulate the problem of learning to rank in a structured setting. Suppose we wish to rank $N$ nodes in the set $\mathcal{V} = \{V_1, \cdots, V_N\}$ and we are given a set of observations $D_1, \cdots, D_T$. Each observation $D_t$ consists of an ordering over the nodes in a subset $\mathcal{V}_t \subseteq \mathcal{V}$, where each node is provided with a corresponding feature vector $\mathbf{x} \in \mathbf{R}^L$ which may be specific to the given observation. The orderings could be provided in the form of ordinal node labels[1], or in the form of pairwise node preferences. The orderings can be represented as a directed graph over the nodes in which a directed edge $e = (V_i \rightarrow V_j)$ is drawn between 2 nodes $V_i, V_j$ iff $V_i$ is preferred to node $V_j$, which we denote as $V_i \succ V_j$. In general, we assume that for any given observation, we observe a partial ordering over nodes, with complete orderings being a special case. We denote the above graph consisting of edges $e = (V_i \rightarrow V_j) \in \mathcal{E}_t$ and the node set $\mathcal{V}_t$ as the order graph $G_t = (\mathcal{V}_t, \mathcal{E}_t)$ for observation $D_t$ so that $D_t = \{G_t, \{\mathbf{x}_n^t\}_{V_n \in \mathcal{V}_t}\}$. A toy example of an observation over 4 nodes is shown in Figure 2(a). Note that under this framework, the absence of an edge between two nodes $V_i, V_j$ in the order graph indicates we cannot assert any preference between the two nodes for the given observation.

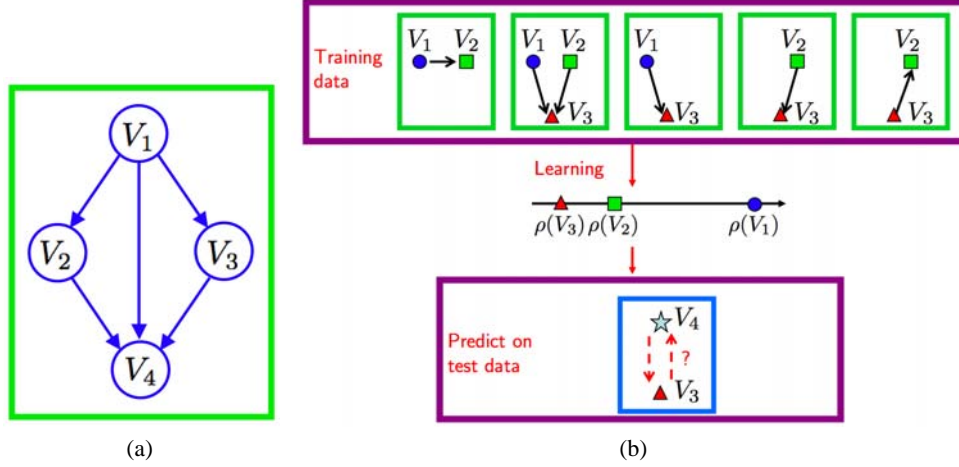

Figure 2: a) An example of an order graph over 4 nodes $V_1, V_2, V_3, V_4$ corresponding to the objects to be ranked. The graph represents the set of preference relationships $V_1 \succ V_2, V_1 \succ V_3, V_1 \succ V_4, V_2 \succ V_4, V_3 \succ V_4$; b) Learning the ranking function from training data. The training data consists of a set of order graphs over subsets of the objects to be ranked. For order graph, the ranking function $\rho$ maps each node to the real line . The goal is to learn $\rho$ such that we minimize our probability of misranking on test observations.

We now define $\rho : \mathcal{V} \rightarrow \mathbf{R}$ as a ranking function which assigns scores to nodes via their feature vectors so that for node $V_i$,

$$S_i = \rho(V_i) + \pi_i \tag{2}$$

where $S_i$ is a scalar and $\pi_i$ is a random variable specific to node $V_i$. We wish to learn such a function given multiple observations $D_1, \cdots, D_T$ so that we minimize the probability of misranking on test observations (Figure 2(b)). The above model allows us to account for the fact that the amount of uncertainty about a node's rank may depend on unobserved features for that node (e.g.: documents associated with certain keywords might have less variability in their rankings than other documents). Under this model, the preference relation $V_i \succ V_j$ is completely equivalent to

$$\rho(V_i) + \pi_i \geq \rho(V_j) + \pi_j \Leftrightarrow \pi_{ij} = \pi_j - \pi_i \leq \rho(V_i) - \rho(V_j). \tag{3}$$

where we have defined $\pi_{ij}$ as a preference variable between nodes $V_i, V_j$.

For each edge $e = (V_i \rightarrow V_j) \in \mathcal{E}_t$ in the order graph, we can define $r(\rho; e, D_t) \equiv \rho(V_i) - \rho(V_j)$ and collect these into the vector $\mathbf{r}(\rho; G_t) \in \mathbf{R}^{|\mathcal{E}_t|}$. Similarly, let $\pi_e \equiv \pi_{ij}$. Having defined the preferences, we must select an appropriate loss measure. A sensible metric here [13] is the joint

probability of observing the order graph $G_t = (\mathcal{V}_t, \mathcal{E}_t)$ corresponding to the partial ordering of nodes in $\mathcal{V}_t$. From Equation (3), this will take the form of a probability measure over events of the type $\pi_e \leq r(\rho; e, D_t)$ so that we obtain

$$Pr\{\mathcal{E}_t | \mathcal{V}_t, \rho\} = Pr\left\{ \bigcap_{e \in \mathcal{E}_t} [\pi_e \leq r(\rho; e, D_t)] \right\} = F_{\boldsymbol{\pi}}(\boldsymbol{r}(\rho; G_t)), \tag{4}$$

where $F_{\boldsymbol{\pi}}$ is the joint CDF over the preference variables $\pi_e$. Given an observation $D_t$, the goal is to learn the ranking function $\rho$ by maximizing Equation (4). Note that under this framework, the set of edges $\mathcal{E}_t$ corresponding to the set of pairwise preferences are treated as random variables which may have a high degree of dependence between one another, so that $F_{\boldsymbol{\pi}}(\boldsymbol{r}(\rho; G_t))$ is a joint CDF over multiple pairwise preferences. The problem of learning the ranking function then consists of scoring multiple nodes simultaneously whilst accounting for dependencies between node scores.

Now, if we are given multiple independent (but not necessarily identically distributed) observations $\mathcal{D} = \{D_1, \cdots, D_T\}$, we can define a *structured loss functional*

$$\mathcal{L}(\rho, F_{\boldsymbol{\pi}}, \mathcal{D}) = -\sum_{t=1}^{T} \log F_{\boldsymbol{\pi}}(\boldsymbol{r}(\rho; G_t)) \tag{5}$$

where each term in the loss functional depends on multiple preference relationships specified by the order graph for observation $t$. The problem of learning then consists of solving the optimization problem

$$\inf_{\rho, F_{\boldsymbol{\pi}}} \mathcal{L}(\rho, F_{\boldsymbol{\pi}}, \mathcal{D}). \tag{6}$$

In general, the above structured loss functional may be difficult to specify, as it takes on the form of a joint CDF over many random variables with a high degree of inter-dependency which may require a large number of parameters to specify. We can, however, compactly represent this using the CDN framework, as we will now show.

### 3.1 Tranforming order graphs into CDNs

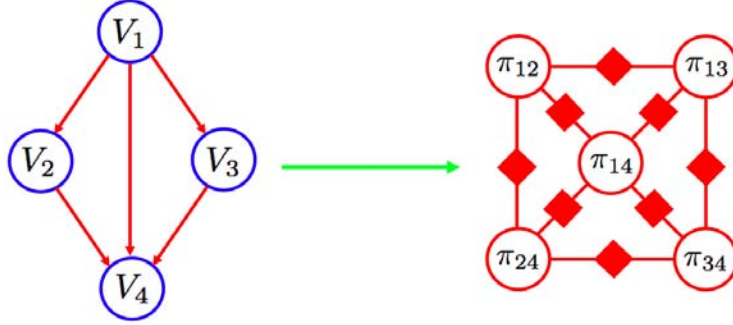

Figure 3: Transforming the order graph $G_t$ into a CDN. For each edge $e = (V_i \rightarrow V_j)$ in the order graph (left), a preference variable $\pi_{ij}$ is created. All such random variables are then connected to one another in a CDN (right), allowing for complex dependencies between preferences.

The representation of the structured loss functional in Equation (5) as a CDN consists of transforming the order graph $G_t$ for each observation into a set of variable nodes in a CDN. More precisely, for each edge $e = (V_i \rightarrow V_j)$ in the order graph, the preference variable $\pi_{ij}$ is created. All such variables are then connected to one another in a CDN (Figure 3), where the pattern of connectivity used will determine the set of dependencies between these preferences $\pi_{ij}$ as given by the marginal and conditional independence properties of CDNs [5]. Thus for any given CDN topology, each preference node $\pi_e$ is a member of some neighborhood of preference nodes $\pi_{e'}$ so that neighboring preferences nodes are marginally dependent of one another.

One possible concern here is that we may require a fully connected CDN topology over all possible pairwise preferences between all nodes in order to capture all of these dependencies, leading to a

model which is cumbersome to learn. In practice, because any observation only conveys information about a small subset of the nodes in $\mathcal{V}$ and because in practice we observe partial orderings between these, the order graph is sparse and so the number of preference nodes in the CDN for the given observation will be much smaller than the worst-case number of all possible pairwise preferences between nodes. Furthermore, we do not have to store a large CDN in memory during training, as we only need to store a single CDN over a relatively small number of preference variables for the current observation. We can thus perform ranking learning in an online fashion by constructing a single CDN for each observation $D_t$ and optimizing the loss $-\log F_{\boldsymbol{\pi}}\big(\boldsymbol{r}(\rho; G_t)\big)$ defined by that CDN for the given observation.

## 4 StructRank: a probabilistic model for structured ranking learning with node labels

Suppose now that each node in the training set is provided with an ordinal node label $y$ along with a feature vector $\mathbf{x}$. For any given order graph over some subset of the nodes, the node labels $y$ allow us to establish edges in the order graph, so that an edge $V_i \rightarrow V_j$ exists between two nodes $V_i, V_j$ iff $y_i > y_j$. We can then parametrically model the ranking function $\rho(V) \equiv \rho(\mathbf{x}; \mathbf{a})$ (where $\mathbf{a}$ is a set of parameters) using a Nadaraya-Watson [10, 12] local estimator with a Gaussian kernel so that

$$\rho(\mathbf{x}; \mathbf{a}) = \frac{\sum_i K(\mathbf{x}_i, \mathbf{x}; \mathbf{a}) y_i}{\sum_i K(\mathbf{x}_i, \mathbf{x}; \mathbf{a})}, \qquad K(\tilde{\mathbf{x}}, \mathbf{x}; \mathbf{a}) = \exp\left(-\frac{1}{2}(\mathbf{x} - \tilde{\mathbf{x}})^T \mathbf{A}(\mathbf{x} - \tilde{\mathbf{x}})\right), \qquad (7)$$

where the summations are taken over all feature vector-label pairs in the training set, with $\mathbf{A} = \mathrm{diag}(a_1^2, \cdots, a_L^2)$. Consider now an edge $e = (V_i \rightarrow V_j)$ in the order graph and define $r_e \equiv r_e(\mathbf{a}; D_t) = \rho(\mathbf{x}_i^t; \mathbf{a}) - \rho(\mathbf{x}_j^t; \mathbf{a})$. For a given order graph, the structured loss functional $\mathcal{L}(\boldsymbol{\theta}; D_t)$ is given by

$$\mathcal{L}(\boldsymbol{\theta}; D_t) = -\log F_{\boldsymbol{\pi}}\big(\boldsymbol{r}(\rho; G_t)\big) = -\sum_{e, e'} \log \phi(r_e(\mathbf{a}; D_t), r_{e'}(\mathbf{a}; D_t)), \qquad (8)$$

where $\boldsymbol{\theta} = \begin{bmatrix} \mathbf{a} & w_1 & w_2 \end{bmatrix}$ is the parameter vector and the function $\phi(r_1, r_2)$ set to a multivariate sigmoidal function so that

$$\phi(r_1, r_2) = \frac{1}{1 + \exp(-w_1 r_1) + \exp(-w_2 r_2)}, \quad w_1, w_2 \geq 0, \qquad (9)$$

where $w_1, w_2$ are weights parameterizing the CDN function $\phi(r_1, r_2)$. It can be readily shown that this choice of CDN function $\phi(r_1, r_2)$, when combined with the constraints $w_1, w_2 > 0$, satisfies all of the necessary and sufficient conditions required for the CDN to represent a valid CDF, as $0 \leq \phi(r_1, r_2) \leq 1$ and is monotonically non-decreasing with respect to all of its arguments. For the given CDN and ranking functions, the learning problem for the current observation $D_t$ then becomes

$$\inf_{\boldsymbol{\theta}} \quad \sum_t \sum_{e, e'} \log\left(1 + \exp\big(-w_1 r_e(\mathbf{a}; D_t)\big) + \exp\big(-w_2 r_{e'}(\mathbf{a}; D_t)\big)\right) \quad \text{s.t.} \quad \boldsymbol{\theta} \geq 0$$

$$\|\boldsymbol{\theta}\|_1 \leq t, \quad (10)$$

where we have introduced a regularizer in the form of an $L_1$-norm constraint. Notice that our model has one parameter per data feature and 2 parameters defining the CDN for any given observation. The gradient $\nabla_{\mathbf{a}} \mathcal{L}(\boldsymbol{\theta}; D_t)$ and the derivatives with respect to the CDN function weights $w_1, w_2$ for a given observation $D_t$ are provided in the Supplementary Information.

## 5 Results

To compare the performance of our proposed framework to other methods, we will use the following three metrics commonly in use in information retrieval research: Precision, Mean Average Precision (MAP) and Normalized Discounted Cumulative Gain (NDCG) [6]. The NDCG accounts for the fact that less relevant documents are less likely to be examine by a user by putting more weight on highly relevant documents than marginally relevant ones.

We downloaded the OHSUMED dataset provided as part of the LETOR 2.0 benchmark [8]. The dataset consists of a set of 106 query-document pairs, with a feature vector and relevance judgment

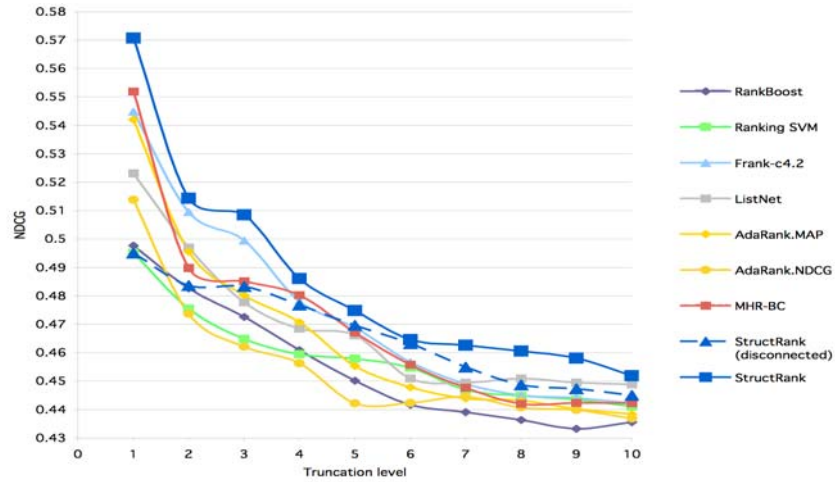

(a)

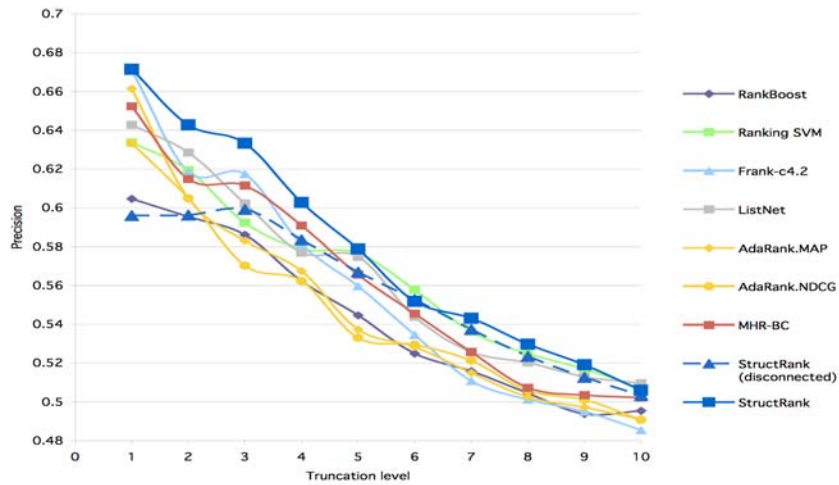

(b)

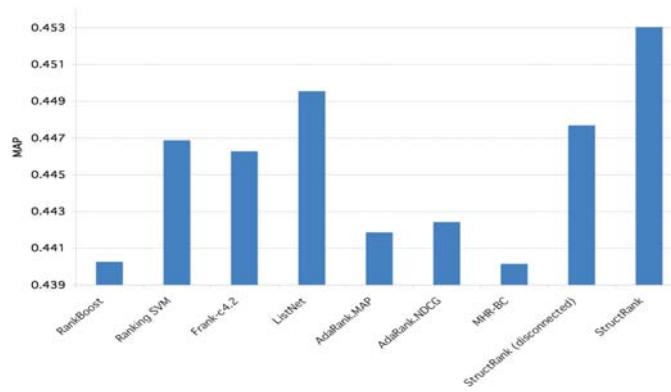

(c)

Figure 4: a) Average NDCG as a function of truncation level $n$ for the OHSUMED dataset. NDCG values are averaged over 5 cross-validation splits; b) Mean average precision (MAP) as a function of truncation level $n$; c) Mean average precision value for several methods.

provided for each pair, where queries correspond to medical searches associated with patient and topic information. There are a total of 16,140 query-document pairs with relevance judgments provided by humans on three ordinal levels: *definitely relevant*, *partially relevant* or *not relevant*. For

any given query, we used the ordinal labels $y$ for each document in the query in order to establish preferences between documents for that query. Each node in the order graph is provided with 25 query-specific features including term frequency, document length, BM25 and LMIR features as well as combinations thereof [1, 11, 14]. In accordance with the nomenclature above, we use the terms query and observation interchangeably.

The OHSUMED dataset is provided in the form of 5 training/validation/test splits of sizes 63/21/22 observations each. To ensure that features are comparable across all observations, we normalized each feature vector within each observation as described in [8]. We performed learning of our model using a constrained stochastic gradients algorithm where for each observation, we prevent updates from violating the inequality constraints in the optimization problem defined by Equation (10) by reducing the learning rate $\alpha$ until the update becomes feasible. We set the default learning rate to $\alpha = 0.5$ and we randomly initialized the model parameters $\mathbf{a}, w_1, w_2$ in the range $[0, 1]$. This optimization was run for 10 epochs (passes through the training set) and $\alpha$ was scaled by $\frac{1}{\sqrt{2}}$ at the end of each epoch. We set the regularization parameter using the validation set for a given data split. Due to the nonconvex nature of the optimization problem, for each cross-validation split, we performed learning using 3 random initializations, and we then selected the model which achieved the best MAP score on the validation set.

We tested a fully connected CDN which models full interdependence between preferences, and a completely disconnected CDN which models preferences independently of one another. The above 3 performance metrics are shown in Figures 4(a),4(b),4(c) in addition to the performances of seven state-of-the-art methods which are part of the LETOR 2.0 benchmarks. At the time of submission, numerical performance scores for ListMLE [13] were not available and so were not included in these plots. With the exception of ListNet and ListMLE, none of the above methods explicitly model dependencies between pairwise preferences. As can be seen, accounting for dependencies between pairwise preferences provides a significant gain in performance compared to modellling preferences as being independent. Additional results on the TREC2004 dataset from LETOR 2.0 are provided in Supplemental Information.

## 6 Discussion

We have proposed here a novel framework for ranking learning using structured loss functionals. We have shown that the problem of learning to rank can be reduced to maximizing a joint CDF over multiple pairwise preferences. We have shown how to compactly represent this using the CDN framework and have applied it to the OHSUMED benchmark dataset. We have demonstrated that representing the dependencies between pairwise preferences leads to improved performance over modelling preferences as being independent of one another.

### 6.1 Relation to RankNet and ListNet/ListMLE

The probability models for ranking proposed by [2, 4, 13] can all be expressed as special cases of models defined by different CDNs. In the case of RankNet [2], the corresponding probability over a given pairwise preference $V_i \succ V_j$ is modelled by a logistic function of $\rho(\mathbf{x}_i) - \rho(\mathbf{x}_j)$ and the model was optimized using cross-entropy loss. The joint probability of preferences can thus be represented as a completely disconnected CDN with logistic functions in which all pairwise object preferences are treated as being independent. In the case of ListNet [4] and ListMLE [13], the probability of observing a complete ordering $V_1 \succ \cdots \succ V_N$ over $N$ objects are defined as products of functions of the type

$$P(V_1 \succ \cdots \succ V_N | D) = \prod_{i=1}^{N} \frac{\exp(\rho(\mathbf{x}_i))}{\sum_{k=i}^{N} \exp(\rho(\mathbf{x}_k))} = \prod_{i=1}^{N} \frac{1}{1 + \sum_{k=i+1}^{N} \exp\left(-\left(\rho(\mathbf{x}_i) - \rho(\mathbf{x}_k)\right)\right)} = \prod_{i=1}^{N} \phi_i(\mathbf{r}_i),$$

which we see is equivalent to a CDN with $N$ multivariate sigmoids. As noted by the authors of [13], the above model is also an example of the Plackett-Luce class of probability models over object scores [9]. In addition, the ListNet/ListMLE frameworks both require a complete ordering over objects by definition: under the CDN framework, we can model partial orderings, with complete orderings as a special case. The connections between RankNet, ListNet and ListMLE and the CDN framework are illustrated in Supplementary Figure 2. Our proposed framework unifies the above

views of ranking as different instantiations of a joint CDF over pairwise preferences and hence as particular types of CDNs. This allows us to consider flexible joint CDFs defined over different subsets of object preferences and over different families of CDN functions so as to capture various data specific properties.

## 6.2  Future directions

Our work here suggests several future directions for research. In [13], it was shown that the log-likelihood corresponding to the probability of an ordering is a good surrogate to the 0-1 loss between the predicted ordering and the true ordering, as the former is differentiable and penalizes mis-orderings in a sensible way. One could investigate connections between the structured loss functionals proposed in this paper and other ranking measures such as NDCG. Another possible direction is to generalize StructRank to products over Gaussian multivariate CDFs or other classes of functions which satisfy the requirements of CDN functions , as in this paper we have elected to use a product of bivariate sigmoids $\phi(r_e, r_{e'})$ to represent our loss functional. Also, it may be fruitful to investigate different CDN topologies: for example, we found that averaging randomly connected CDNs are very fast to learn and perform comparably to the fully-connected CDN we used in this paper (data not shown). In addition, we have only investigated representing the loss functional using a single CDN function: this could easily be generalized to $K$ functions. Lastly, alternatives to the Nadaraya-Watson local estimator, such as the neural networks used in [2, 4, 13], can be investigated.

## Footnotes

[1]It is crucial to note that node labels may in general not be directly comparable with one another from one observation to the next (e.g.: documents with the same rating might not truly have the same degree of relevance for different queries), or the scale of the labels may be arbitrary.

## References

[1] R. Baeza-Yates and B. Ribeiro-Neto. Modern information retrieval. *Addison Wesley*, 1999.

[2] C.J.C. Burges, T. Shaked, E. Renshaw, A. Lazier, M. Deeds, N. Hamilton and G. Hullender. Learning to rank using gradient descent. *In Proceedings of the Twenty-Second International Conference on Machine Learning (ICML)*, 2005.

[3] C.J.C. Burges, R. Ragno and Q.V. Le. Learning to rank with nonsmooth cost functions. *In Proceedings of the Nineteenth Annual Conference on Neural Information Processing Systems (NIPS)*, 2007.

[4] Z. Cao, T. Qin, T.Y. Liu, M.F. Tsai and H. Li. Learning to rank: from pairwise approach to listwise approach. *In Proceedings of the Twenty-Fourth International Conference on Machine Learning (ICML)*, 2007.

[5] J.C. Huang and B.J. Frey. Cumulative distribution networks and the derivative-sum-product algorithm. *In Proceedings of the Twenty-Fourth Conference on Uncertainty in Artificial Intelligence (UAI)*, 2008.

[6] K. Jarvelin and J. Kekalainen. Cumulated evaluation of IR techniques, *ACM Information Systems*, 2002.

[7] T. Joachims. A support vector method for multivariate performance measures. *In Proceedings of the Twenty-Second International Conference on Machine Learning (ICML)*, 2005.

[8] T.Y. Liu, J. Xu, T. Qin, W. Xiong and H. Li. LETOR: Benchmark dataset for research on learning to rank for information retrieval. *LR4IR 2007, in conjunction with SIGIR 2007*, 2007.

[9] J. I. Marden. Analyzing and modeling rank data. *CRC Press*, 1995.

[10] E.A. Nadaraya. On estimating regression. *Theory of Probability and its Applications* **9(1)**, pp. 141-142, 1964.

[11] S.E. Robertson. Overview of the OKAPI projects. *Journal of Documentation* **53 (1)**, pp. 3-7, 1997.

[12] G.S. Watson. Smooth regression analysis. *The Indian Journal of Statistics. Series A* **26**, pp. 359-372, 1964.

[13] F. Xia, T.Y. Liu, J. Wang, W. Zhang and H. Li. Listwise approach to learning to rank - theory and algorithm. *In Proceedings of the Twenty-Fifth International Conference on Machine Learning (ICML)*, 2008.

[14] C. Zhai and J. Lafferty. A study of smoothing methods for language models applied to ad hoc information retrieval. *In Proceedings of SIGIR 2001*, 2001.
